# Learning, Regularization and Ill-Posed Inverse Problems

**Lorenzo Rosasco**
DISI, Università di Genova
Genova, I
rosasco@disi.unige.it

**Andrea Caponnetto**
DISI, Università di Genova
Genova, I
caponnetto@disi.unige.it

**Ernesto De Vito**
Dipartimento di Matematica
Università di Modena
and INFN, Sezione di Genova
Genova, I
devito@unimo.it

**Umberto De Giovannini**
DISI, Università di Genova
Genova, I
umberto.degiovannini@fastwebnet.it

**Francesca Odone**
DISI, Università di Genova
Genova, I
odone@disi.unige.it

## Abstract

Many works have shown that strong connections relate learning from examples to regularization techniques for ill-posed inverse problems. Nevertheless by now there was no formal evidence neither that learning from examples could be seen as an inverse problem nor that theoretical results in learning theory could be independently derived using tools from regularization theory. In this paper we provide a positive answer to both questions. Indeed, considering the square loss, we translate the learning problem in the language of regularization theory and show that consistency results and optimal regularization parameter choice can be derived by the discretization of the corresponding inverse problem.

## 1 Introduction

The main goal of learning from examples is to infer an estimator, given a finite sample of data drawn according to a fixed but unknown probabilistic input-output relation. The desired property of the selected estimator is to perform well on new data, i.e. it should generalize. The fundamental works of Vapnik and further developments [16], [8], [5], show that the key to obtain a meaningful solution to the above problem is to control the complexity of the solution space. Interestingly, as noted by [12], [8], [2], this is the idea underlying regularization techniques for ill-posed inverse problems [15], [7]. In such a context to avoid undesired oscillating behavior of the solution we have to restrict the solution space.

Not surprisingly the form of the algorithms proposed in both theories is strikingly similar. Anyway a careful analysis shows that a rigorous connection between learning and regularization for inverse problem is not straightforward. In this paper we consider the square loss and show that the problem of learning can be translated into a convenient inverse problem and consistency results can be derived in a general setting. When a generic loss is considered the analysis becomes immediately more complicated.

Some previous works on this subject considered the special case in which the elements of the input space are fixed and not probabilistically drawn [11], [9]. Some weaker results in the same spirit of those presented in this paper can be found in [13] where anyway the connections with inverse problems is not discussed. Finally, our analysis is close to the idea of stochastic inverse problems discussed in [16]. It follows the plan of the paper. After recalling the main concepts and notation of learning and inverse problems, in section 4 we develop a formal connection between the two theories. In section 5 the main results are stated and discussed. Finally in section 6 we conclude with some remarks and open problems.

## 2   Learning from examples

We briefly recall some basic concepts of learning theory [16], [8]. In the framework of learning, there are two sets of variables: the input space $X$, compact subset of $\mathbf{R}^n$, and the output space $Y$, compact subset of $\mathbf{R}$. The relation between the input $x \in X$ and the output $y \in Y$ is described by a probability distribution $\rho(x,y) = \nu(x)\rho(y|x)$ on $X \times Y$. The distribution $\rho$ is known only through a sample $\mathbf{z} = (\mathbf{x}, \mathbf{y}) = ((x_1, y_1), \ldots, (x_\ell, y_\ell))$, called *training set*, drawn i.i.d. according to $\rho$. The goal of learning is, given the sample $\mathbf{z}$, to find a function $f_{\mathbf{z}} : X \to \mathbf{R}$ such that $f_{\mathbf{z}}(x)$ is an *estimate* of the output $y$ when the new input $x$ is given. The function $f_{\mathbf{z}}$ is called *estimator* and the rule that, given a sample $\mathbf{z}$, provides us with $f_{\mathbf{z}}$ is called *learning algorithm*.

Given a measurable function $f : X \to \mathbf{R}$, the ability of $f$ to describe the distribution $\rho$ is measured by its expected risk defined as

$$I[f] = \int_{X \times Y} (f(x) - y)^2 \, d\rho(x,y).$$

The regression function

$$g(x) = \int_Y y \, d\rho(y|x),$$

is the minimizer of the expected risk over the set of all measurable functions and always exists since $Y$ is compact. Usually, the regression function cannot be reconstructed exactly since we are given only a finite, possibly small, set of examples $\mathbf{z}$.

To overcome this problem, in the regularized least squares algorithm an hypothesis space $\mathcal{H}$ is fixed, and, given $\lambda > 0$, an estimator $f_{\mathbf{z}}{}^\lambda$ is defined as the solution of the regularized least squares problem,

$$\min_{f \in \mathcal{H}} \{ \frac{1}{\ell} \sum_{i=1}^{\ell} (f(x_i) - y_i)^2 + \lambda \|f\|_{\mathcal{H}}^2 \}. \tag{1}$$

The regularization parameter $\lambda$ has to be chosen depending on the available data, $\lambda = \lambda(\ell, \mathbf{z})$, in such a way that, for every $\epsilon > 0$

$$\lim_{\ell \to +\infty} \mathrm{P} \left[ I[f_{\mathbf{z}}{}^{\lambda(\ell, \mathbf{z})}] - \inf_{f \in \mathcal{H}} I[f] \geq \epsilon \right] = 0. \tag{2}$$

We note that in general $\inf_{f \in \mathcal{H}} I[f]$ is larger that $I[g]$ and represents a sort of irreducible error associated with the choice of the space $\mathcal{H}$. The above convergence in probability is usually called *consistency* of the algorithm [16] [14].

## 3  Ill-Posed Inverse Problems and Regularization

In this section we give a very brief account of linear inverse problems and regularization theory [15], [7]. Let $\mathcal{H}$ and $\mathcal{K}$ be two Hilbert spaces and $A : \mathcal{H} \to \mathcal{K}$ a linear bounded operator. Consider the equation

$$Af = g_\delta \tag{3}$$

where $g_\delta, g \in \mathcal{K}$ and $\|g - g_\delta\|_{\mathcal{K}} \leq \delta$. Here $g$ represents the exact, unknown data and $g_\delta$ the available, noisy data. Finding the function $f$ satisfying the above equation, given $A$ and $g_\delta$, is the linear inverse problem associated to Eq. (3). The above problem is, in general, ill-posed, that is, the Uniqueness can be restored introducing the Moore-Penrose generalized inverse $f^\dagger = A^\dagger g$ defined as the minimum norm solution of the problem

$$\min_{f \in \mathcal{H}} \|Af - g\|_{\mathcal{K}}^2 . \tag{4}$$

However the operator $A^\dagger$ is usually not bounded so, in order to ensure a continuous dependence of the solution on the data, the following Tikhonov regularization scheme can be considered[1]

$$\min_{f \in \mathcal{H}} \{\|Af - g_\delta\|_{\mathcal{K}}^2 + \lambda \|f\|_{\mathcal{H}}^2\}, \tag{5}$$

whose unique minimizer is given by

$$f_\delta^\lambda = (A^*A + \lambda I)^{-1} A^* g_\delta, \tag{6}$$

where $A^*$ denotes the adjoint of $A$.

A crucial step in the above algorithm is the choice of the regularization parameter $\lambda = \lambda(\delta, g_\delta)$, as a function of the noise level $\delta$ and the data $g_\delta$, in such a way that

$$\lim_{\delta \to 0} \left\| f_\delta^{\lambda(\delta, g_\delta)} - f^\dagger \right\|_{\mathcal{H}} = 0, \tag{7}$$

that is, the regularized solution $f_\delta^{\lambda(\delta, g_\delta)}$ converges to the generalized solution $f^\dagger = A^\dagger g$ ($f^\dagger$ exists if and only if $Pg \in \mathrm{Range}(A)$, where $P$ is the projection on the closure of the range of $A$ and, in that case, $Af^\dagger = Pg$) when the noise $\delta$ goes to zero.

The similarity between regularized least squares algorithm (1) and Tikhonov regularization (5) is apparent. However, several difficulties emerge. First, to treat the problem of learning in the setting of ill-posed inverse problems we have to define a direct problem by means of a suitable operator $A$. Second, in the context of learning, it is not clear the nature of the noise $\delta$. Finally we have to clarify the relation between consistency (2) and the kind of convergence expressed by (7). In the following sections we will show a possible way to tackle these problems.

## 4  Learning as an Inverse Problem

We can now show how the problem of learning can be rephrased in a framework close to the one presented in the previous section.
We assume that hypothesis space $\mathcal{H}$ is a reproducing kernel Hilbert space [1] with a continuous kernel $K : X \times X \to \mathbf{R}$. If $x \in X$, we let $K_x(s) = K(s, x)$, and, if $\nu$ is the marginal distribution of $\rho$ on $X$, we define the bounded linear operator $A : \mathcal{H} \to L^2(X, \nu)$ as

$$(Af)(x) = \langle f, K_x \rangle_{\mathcal{H}} = f(x),$$

that is, $A$ is the canonical injection of $\mathcal{H}$ in $L^2(X, \nu)$. In particular, for all $f \in \mathcal{H}$, the expected risk becomes,

$$I[f] = \|Af - g\|^2_{L^2(X,\nu)} + I[g],$$

where $g$ is the regression function [2]. The above equation clarifies that if the expected risk admits a minimizer $f_{\mathcal{H}}$ on the hypothesis space $\mathcal{H}$, then it is exactly the generalized solution[2] $f^\dagger = A^\dagger g$ of the problem

$$Af = g. \tag{8}$$

Moreover, given a training set $\mathbf{z} = (\mathbf{x}, \mathbf{y})$, we get a discretized version $A_{\mathbf{x}} : \mathcal{H} \to \mathbf{E}^\ell$ of $A$, that is

$$(A_{\mathbf{x}} f)_i = \langle f, K_{x_i} \rangle_{\mathcal{H}} = f(x_i),$$

where $\mathbf{E}^\ell = \mathbf{R}^\ell$ is the finite dimensional euclidean space endowed with the scalar product

$$\langle \mathbf{y}, \mathbf{y}' \rangle_{\mathbf{E}^\ell} = \frac{1}{\ell} \sum_{i=1}^{\ell} y_i y_i'.$$

It is straightforward to check that

$$\frac{1}{\ell} \sum_{i=1}^{\ell} (f(x_i) - y_i)^2 = \|A_{\mathbf{x}} f - \mathbf{y}\|^2_{\mathbf{E}^\ell},$$

so that the estimator $f_{\mathbf{z}}{}^\lambda$ given by the regularized least squares algorithm is the regularized solution of the discrete problem

$$A_{\mathbf{x}} f = \mathbf{y}. \tag{9}$$

At this point it is useful to remark the following two facts. First, in learning from examples we are not interested into finding an approximation of the generalized solution of the discretized problem (9), but we want to find a stable approximation of the solution of the exact problem (8) (compare with [9]). Second, we notice that in learning theory the consistency property (2) involves the control of the quantity

$$I[f_{\mathbf{z}}{}^\lambda] - \inf_{f \in \mathcal{H}} I[f] = \|Af - g\|^2_{L^2(X,\nu)} - \inf_{f \in \mathcal{H}} \|Af - g\|^2_{L^2(X,\nu)}. \tag{10}$$

If $P$ is the projection on the closure of the range of $A$, the definition of $P$ gives

$$I[f_{\mathbf{z}}{}^\lambda] - \inf_{f \in \mathcal{H}} I[f] = \left\| Af_{\mathbf{z}}{}^\lambda - Pg \right\|^2_{L^2(X,\nu)} \tag{11}$$

(the above equality stronlgy depends on the fact that the loss function is the square loss). In the inverse problem setting, the square root of the above quantity is called the *residue* of the solution $f_{\mathbf{z}}{}^\lambda$. Hence, consistency is controlled by the residue of the estimator, instead of the reconstruction error $\left\| f_{\mathbf{z}}{}^\lambda - f^\dagger \right\|_{\mathcal{H}}$ (as in inverse problems). In particular, consistency is a weaker condition than the one required by (7) and does not require the existence of the generalized solution $f_{\mathcal{H}}$.

## 5 Regularization, Stochastic Noise and Consistency

To apply the framework of ill-posed inverse problems of Section 3 to the formulation of learning proposed above, we note that the operator $A_{\mathbf{x}}$ in the discretized problem (9) differs from the operator $A$ in the exact problem (8) and a measure of the difference between $A_{\mathbf{x}}$ and $A$ is required. Moreover, the noisy data $\mathbf{y} \in \mathbf{E}^\ell$ and the exact data $g \in L^2(X, \nu)$ belong to different spaces, so that the notion of noise has to be modified. Given the above premise our derivation of consistency results is developed in two steps: we first study the residue of the solution by means of a measure of the noise due to discretization and then we show a possible way to give a probabilistic evaluation of the noise previously introduced.

### 5.1 Bounding the Residue of the Regularized Solution

We recall that the regularized solutions of problems (9) and (8) are given by

$$
\begin{aligned}
f_{\mathbf{z}}^{\lambda} &= (A_{\mathbf{x}}^* A_{\mathbf{x}} + \lambda I)^{-1} A_{\mathbf{x}}^* \mathbf{y}, \\
f^{\lambda} &= (A^* A + \lambda I)^{-1} A^* g.
\end{aligned}
$$

The above equations show that $f_{\mathbf{z}}^{\lambda}$ and $f^{\lambda}$ depend only on $A_{\mathbf{x}}^* A_{\mathbf{x}}$ and $A^* A$ which are operators from $\mathcal{H}$ into $\mathcal{H}$ and on $A_{\mathbf{x}}^* \mathbf{y}$ and $A^* g$ which are elements of $\mathcal{H}$, so that the space $\mathbf{E}^{\ell}$ disappears. This observation suggests that noise levels could be $\|A_{\mathbf{x}}^* A_{\mathbf{x}} - A^* A\|_{\mathcal{L}(\mathcal{H})}$ and $\|A_{\mathbf{x}}^* \mathbf{y} - A^* g\|_{\mathcal{H}}$, where $\|\cdot\|_{\mathcal{L}(\mathcal{H})}$ is the uniform operator norm. To this purpose, for every $\delta = (\delta_1, \delta_2) \in \mathbf{R}_+^2$ we define the collection of training sets.

$$
\mathcal{U}_{\delta} = \{\mathbf{z} \in (X \times Y)^{\ell} \mid \|A_{\mathbf{x}}^* \mathbf{y} - A^* g\|_{\mathcal{H}} \leq \delta_1, \ \|A_{\mathbf{x}}^* A_{\mathbf{x}} - A^* A\|_{\mathcal{L}(\mathcal{H})} \leq \delta_2, \ \ell \in \mathbf{N}\}
$$

and we let $M = \sup\{|y| \mid y \in Y\}$. The next theorem is the central result of the paper.

**Theorem 1** *If $\lambda > 0$, the following inequalities hold*

*1. for any training set $\mathbf{z} \in \mathcal{U}_{\delta}$*

$$
\left| \left\|Af_{\mathbf{z}}^{\lambda} - Pg\right\|_{L^2(X,\nu)} - \left\|Af^{\lambda} - Pg\right\|_{L^2(X,\nu)} \right| \leq \frac{M\delta_2}{4\lambda} + \frac{\delta_1}{2\sqrt{\lambda}}
$$

*2. if $Pg \in \mathrm{Range}(A)$, for any training set $\mathbf{z} \in \mathcal{U}_{\delta}$,*

$$
\left| \left\|f_{\mathbf{z}}^{\lambda} - f^{\dagger}\right\|_{\mathcal{H}} - \left\|f^{\lambda} - f^{\dagger}\right\|_{\mathcal{H}} \right| \leq \frac{M\delta_2}{2\lambda^{\frac{3}{2}}} + \frac{\delta_1}{\lambda}
$$

*Moreover if we choose $\lambda = \lambda(\delta, \mathbf{z})$ in such a way that*

$$
\begin{cases}
\lim_{\delta \to 0} \ \sup_{\mathbf{z} \in \mathcal{U}_{\delta}} \ \lambda(\delta, \mathbf{z}) &= 0 \\
\lim_{\delta \to 0} \ \sup_{\mathbf{z} \in \mathcal{U}_{\delta}} \ \frac{\delta_1^2}{\lambda(\delta, \mathbf{z})} &= 0 \\
\lim_{\delta \to 0} \ \sup_{\mathbf{z} \in \mathcal{U}_{\delta}} \ \frac{\delta_2}{\lambda(\delta, \mathbf{z})} &= 0
\end{cases}
\tag{12}
$$

*then*

$$
\lim_{\delta \to 0} \ \sup_{\mathbf{z} \in \mathcal{U}_{\delta}} \left\|Af_{\mathbf{z}}^{\lambda(\delta, \mathbf{z})} - Pg\right\|_{L^2(X,\nu)} = 0.
\tag{13}
$$

We omit the complete proof and refer to [3]. Briefly, the idea is to note that

$$
\left| \left\|Af_{\mathbf{z}}^{\lambda} - Pg\right\|_{L^2(X,\nu)} - \left\|Af^{\lambda} - Pg\right\|_{L^2(X,\nu)} \right|
$$
$$
\leq \left\|Af_{\mathbf{z}}^{\lambda} - Af^{\lambda}\right\|_{L^2(X,\nu)} = \left\|(A^* A)^{\frac{1}{2}}(f_{\mathbf{z}}^{\lambda} - f^{\lambda})\right\|_{\mathcal{H}}
$$

where the last equation follows by polar decomposition of the operator $A$. Moreover a simple algebraic computation gives

$$
f_{\mathbf{z}}^{\lambda} - f^{\lambda} = (A^* A + \lambda I)^{-1}(A^* A - A_{\mathbf{x}}^* A_{\mathbf{x}})(A_{\mathbf{x}}^* A_{\mathbf{x}} + \lambda I)^{-1} A_{\mathbf{x}}^* \mathbf{y} + (A^* A + \lambda I)^{-1}(A_{\mathbf{x}}^* \mathbf{y} - A^* g)
$$

where the relevant quantities for definition of the noise appear.

The first item in the above proposition quantifies the difference between the residues of the regularized solutions of the exact and discretized problems in terms of the noise level $\delta = (\delta_1, \delta_2)$. As mentioned before this is exactly the kind of result needed to derive consistency. On the other hand the last part of the proposition gives sufficient conditions on the parameter $\lambda$ to ensure convergence of the residue to zero as the level noise decreases. The above results were obtained introducing the collection $\mathcal{U}_{\delta}$ of training sets compatible with a certain noise level $\delta$. It is left to quantify the noise level corresponding to a training set of cardinality $\ell$. This will be achieved in a probabilistic setting in the next section.

## 5.2 Stochastic Evaluation of the Noise

In this section we estimate the discretization noise $\delta = (\delta_1, \delta_2)$.

**Theorem 2** *Let $\epsilon_1, \epsilon_2 > 0$ and $\kappa = \sup_{x \in X} \sqrt{K(x,x)}$, then*

$$\mathrm{P}\left[ \|A^* g - A_{\mathbf{x}}{}^* \mathbf{y}\|_{\mathcal{H}} \le \frac{M\kappa}{\sqrt{\ell}} + \epsilon_1,\ \|A^* A - A_{\mathbf{x}}{}^* A_{\mathbf{x}}\|_{\mathcal{L}(\mathcal{H})} \le \frac{\kappa^2}{\sqrt{\ell}} + \epsilon_2 \right]$$

$$\ge 1 - e^{-\frac{\epsilon_1^2 \ell}{2\kappa^2 M^2}} - e^{-\frac{\epsilon_2^2 \ell}{2\kappa^4}} \tag{14}$$

The proof is given in [3] and it is based on McDiarmid inequality [10] applied to the random variables

$$F(\mathbf{z}) = \|A_{\mathbf{x}}{}^* \mathbf{y} - A^* g\|_{\mathcal{H}} \quad G(\mathbf{z}) = \|A_{\mathbf{x}}{}^* A_{\mathbf{x}} - A^* A\|_{\mathcal{L}(\mathcal{H})}.$$

Other estimates of the noise $\delta$ can be given using, for example, union bounds and Hoeffding's inequality. Anyway rather then providing a tight analysis our concern was to find an natural, explicit and easy to prove estimate of $\delta$.

## 5.3 Consistency and Regularization Parameter Choice

Combining Theorems 1 and 2, we easily derive the following corollary.

**Corollary 1** *Given $0 < \eta < 1$, with probability greater that $1 - \eta$,*

$$\left| \left\| A f_{\mathbf{z}}^{\lambda} - Pg \right\|_{L^2(X,\nu)} - \left\| A f^{\lambda} - Pg \right\|_{L^2(X,\nu)} \right|$$

$$\le \frac{\kappa M}{2\sqrt{\ell}} \left( \frac{1}{\sqrt{\lambda}} + \frac{\kappa}{2\lambda} \right) \left( 1 + \log \sqrt{\frac{4}{\eta}} \right) \tag{15}$$

*for all $\lambda > 0$.*

Recalling (10) and (11) it is straightforward to check that the above inequality can be easily restated in the usual learning notation, in fact we obtain

$$I[f_{\mathbf{z}}^{\lambda}] \le \left[ \underbrace{\frac{\kappa L}{2\sqrt{\ell}} \left( \frac{1}{\sqrt{\lambda}} + \frac{\kappa}{2\lambda} \right) \left( 1 + \log \sqrt{\frac{4}{\eta}} \right)}_{\text{sample error}} + \underbrace{\left\| A f^{\lambda} - Pg \right\|_{L^2(X,\nu)}}_{\text{approximation error}} \right]^2 + \underbrace{\inf_{f \in \mathcal{H}} I[f]}_{\text{irreducible error}} \quad .$$

In the above inequality the first term plays the role of sample error. If we choose the regularization parameter so that $\lambda = \lambda(\ell, \mathbf{z}) = O(\frac{1}{\ell^b})$, with $0 < b < \frac{1}{2}$ the sample error converges in probability to zero with order $O\left( \sqrt{\frac{1}{\ell^{1-2b}}} \right)$ when $\ell \to \infty$. On the other hand the second term represents the approximation error and it is possible to show, using standard results from spectral theory, that it vanishes as $\lambda$ goes to zero [7]. Finally, the last term represents the minimum attainable risk once the hypothesis space $\mathcal{H}$ has been chosen.

From the above observations it is clear that consistency is ensured once the parameter $\lambda$ is chosen according to the aforementioned conditions. Nonetheless to provide convergence rates it is necessary to control the convergence rate of the approximation error. Unfortunately it is well known that this can be accomplished only making some assumptions on the underlying probability distribution $\rho$ (see for example [2]). It can be shown that if the explicit dependence of the approximation error on $\lambda$ is not available we cannot determine

an optimal a priori (data independent) dependency $\lambda = \lambda(\ell)$ for the regularization parameter. Nevertheless a posteriori (data dependent) choices $\lambda = \lambda(\ell, \mathbf{z})$ can be considered to automatically achieve optimal convergence rate [5], [6]. With respect to this last fact we notice that the set of samples such that inequality (14) holds depends on $\ell$ and $\eta$, but does not depend $\lambda$, so that we can consider without any further effort a posteriori parameter choices (compare with [4], [5]).

Finally, we notice that the estimate (15) is the result of two different procedures: Theorem 1, which is of functional type, gives the dependence of the bound by the regularization parameter $\lambda$ and by the noise levels $\|A_{\mathbf{x}}^* A_{\mathbf{x}} - A^* A\|_{\mathcal{L}(\mathcal{H})}$ and $\|A_{\mathbf{x}}^* \mathbf{y} - A^* g\|_{\mathcal{H}}$, whereas Theorem 2, which is of probabilistic nature, relates the noise levels to the number of data $\ell$ and the confidence level $\eta$.

## 6 Conclusions

In this paper we defined a direct and inverse problem suitable for the learning problem and derived consistency results for the regularized least squares algorithm. Though our analysis formally explains the connections between learning theory and linear inverse problems, its main limit is that we considered only the square loss. We briefly sketch how the arguments presented in the paper extend to general loss functions. For sake of simplicity we consider a differentiable loss function $V$. It is easy to see that the minimizer $f_{\mathcal{H}}$ of the expected risk satisfies the following equation

$$S f_{\mathcal{H}} = 0 \tag{16}$$

where $S = L_K \circ O$ and $L_K$ is the integral operator with kernel K, that is

$$(L_K f)(x) = \int_X K(x, s) f(s) d\nu(s)$$

and $O$ is the operator defined by

$$(Of)(x) = \int_Y V'(y, f(x)) d\rho(y|x).$$

If we consider a generic differentiable loss the operator $O$ and hence $S$ is non linear, and estimating $f_{\mathcal{H}}$ is an ill-posed non linear inverse problem. It is well known that the theory for this kind of problems is much less developed than the corresponding theory for linear problems. Moreover, since, in general, $I[f]$ does not define a metric, it is not so clear the relation between the expected risk and the residue. It appears evident that the attempt to extend our results to a wider class of loss function is not straightforward. A possible way to tackle the problem, further developing our analysis, might pass through the exploitation of a natural convexity assumption on the loss function. Future work also aims to derive tighter probabilistic bounds on the noise using recently proposed data dependent techniques.

### Acknowledgments

We would like to thank M.Bertero, C. De Mol, M. Piana, T. Poggio, G. Talenti, A. Verri for useful discussions and suggestions. This research has been partially funded by the INFM Project MAIA, the FIRB Project ASTA[2] and the IST Programme of the European Community, under the PASCAL Network of Excellence, IST-2002-506778.

## Footnotes

[1]In the framework of inverse problems, many other regularization procedures are introduced [7]. For simplicity we only treat the Tikhonov regularization.

[2]The fact that $f_{\mathcal{H}}$ is the minimal norm solution of (4) is ensured by the assumption that the support of the measure $\nu$ is $X$, since in this case the operator $A$ is injective.

## References

[1] N. Aronszajn. Theory of reproducing kernels. *Trans. Amer. Math. Soc.*, 68:337–404, 1950.

[2] Felipe Cucker and Steve Smale. On the mathematical foundations of learning. *Bull. Amer. Math. Soc. (N.S.)*, 39(1):1–49 (electronic), 2002.

[3] E. De Vito, A. Caponnetto, and L. Rosasco. Discretization error analysis for Tikhonov regularization. *submitted to Inverse Problem*, 2004. available http://www.disi.unige.it/person/RosascoL/publications/discre_iop.pdf.

[4] E. De Vito, A. Caponnetto, and L. Rosasco. Model selection for regularized least-squares algorithm in learning theory. *to appear on Journal Machine Learning Research*, 2004.

[5] L. Devroye, L. Györfi, and G. Lugosi. *A Probabilistic Theory of Pattern Recognition*. Number 31 in Applications of mathematics. Springer, New York, 1996.

[6] Schock E. and Sergei V. Pereverzev. On the adaptive selection of the parameter in regularization of ill-posed problems. Technical report, University of Kaiserslautern, august 200r.

[7] Heinz W. Engl, Martin Hanke, and Andreas Neubauer. *Regularization of inverse problems*, volume 375 of *Mathematics and its Applications*. Kluwer Academic Publishers Group, Dordrecht, 1996.

[8] Theodoros Evgeniou, Massimiliano Pontil, and Tomaso Poggio. Regularization networks and support vector machines. *Adv. Comput. Math.*, 13(1):1–50, 2000.

[9] Vera Kurkova. Supervised learning as an inverse problem. Technical Report 960, Institute of Computer Science, Academy of Sciences of the Czech Republic, April 2004.

[10] Colin McDiarmid. On the method of bounded differences. In *Surveys in combinatorics, 1989 (Norwich, 1989)*, volume 141 of *London Math. Soc. Lecture Note Ser.*, pages 148–188. Cambridge Univ. Press, Cambridge, 1989.

[11] S. Mukherjee, T. Niyogi, P.and Poggio, and R. Rifkin. Statistical learning: Stability is sufficient for generalization and necessary and sufficient for consistency of empirical risk minimization. Technical Report CBCL Paper 223, Massachusetts Institute of Technology, january revision 2004.

[12] T. Poggio and Girosi F. Networks for approximation and learning. *Proc. IEEE*, 78:1481–1497, 1990.

[13] Cynthia Rudin. A different type of convergence for statistical learning algorithms. Technical report, Program in Applied and Computational Mathematics Princeton University, 2004.

[14] I. Steinwart. Consistency of support vector machines and other regularized kernel machines. *IEEE Transaction on Information Theory*, 2004. (accepted).

[15] Andrey N. Tikhonov and Vasiliy Y. Arsenin. *Solutions of ill-posed problems*. V. H. Winston & Sons, Washington, D.C.: John Wiley & Sons, New York, 1977. Translated from the Russian, Preface by translation editor Fritz John, Scripta Series in Mathematics.

[16] Vladimir N. Vapnik. *Statistical learning theory*. Adaptive and Learning Systems for Signal Processing, Communications, and Control. John Wiley & Sons Inc., New York, 1998. A Wiley-Interscience Publication.
